# Localizing 3D cuboids in single-view images

**Jianxiong Xiao**    **Bryan C. Russell**\*    **Antonio Torralba**

Massachusetts Institute of Technology    \*University of Washington

## Abstract

In this paper we seek to detect rectangular cuboids and localize their corners in uncalibrated single-view images depicting everyday scenes. In contrast to recent approaches that rely on detecting vanishing points of the scene and grouping line segments to form cuboids, we build a discriminative parts-based detector that models the appearance of the cuboid corners and internal edges while enforcing consistency to a 3D cuboid model. Our model copes with different 3D viewpoints and aspect ratios and is able to detect cuboids across many different object categories. We introduce a database of images with cuboid annotations that spans a variety of indoor and outdoor scenes and show qualitative and quantitative results on our collected database. Our model out-performs baseline detectors that use 2D constraints alone on the task of localizing cuboid corners.

## 1   Introduction

Extracting a 3D representation from a single-view image depicting a 3D object has been a long-standing goal of computer vision [20]. Traditional approaches have sought to recover 3D properties, such as creases, folds, and occlusions of surfaces, from a line representation extracted from the image [18]. Among these are works that have characterized and detected *geometric primitives*, such as quadrics (or "geons") and surfaces of revolution, which have been thought to form the components for many different object types [1]. While these approaches have achieved notable early successes, they could not be scaled-up due to their dependence on reliable contour extraction from natural images.

In this work we focus on the task of detecting *rectangular cuboids*, which are a basic geometric primitive type and occur often in 3D scenes (e.g. indoor and outdoor man-made scenes [22, 23, 24]). Moreover, we wish to recover the shape parameters of the detected cuboids. The detection and recovery of shape parameters yield at least a partial geometric description of the depicted scene, which allows a system to reason about the affordances of a scene in an object-agnostic fashion [9, 15]. This is especially important when the category of the object is ambiguous or unknown.

There have been several recent efforts that revisit this problem [9, 11, 12, 17, 19, 21, 26, 28, 29]. Although there are many technical differences amongst these works, the main pipeline of these approaches is similar. Most of them estimate the camera parameters using three orthogonal vanishing points with a Manhattan world assumption of a man-made scene. They detect line segments via Canny edges and recover surface orientations [13] to form 3D cuboid hypotheses using bottom-up grouping of line and region segments. Then, they score these hypotheses based on the image evidence for lines and surface orientations [13].

In this paper we look to take a different approach for this problem. As shown in Figure 1, we aim to build a 3D cuboid detector to detect individual boxy volumetric structures. We build a discriminative parts-based detector that models the appearance of the corners and internal edges of cuboids while enforcing spatial consistency of the corners and edges to a 3D cuboid model. Our model is trained in a similar fashion to recent work that detects articulated human body joints [27].

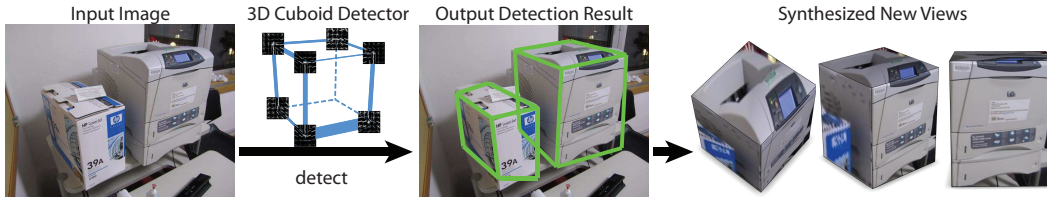

Figure 1: Problem summary. Given a single-view input image, our goal is to detect the 2D corner locations of the cuboids depicted in the image. With the output part locations we can subsequently recover information about the camera and 3D shape via camera resectioning.

Our cuboid detector is trained across different 3D viewpoints and aspect ratios. This is in contrast to view-based approaches for object detection that train separate models for different viewpoints, e.g. [7]. Moreover, instead of relying on edge detection and grouping to form an initial hypothesis of a cuboid [9, 17, 26, 29], we use a 2D sliding window approach to exhaustively evaluate all possible detection windows. Also, our model does not rely on any preprocessing step, such as computing surface orientations [13]. Instead, we learn the parameters for our model using a structural SVM framework. This allows the detector to adapt to the training data to identify the relative importance of corners, edges and 3D shape constraints by learning the weights for these terms. We introduce an annotated database of images with geometric primitives labeled and validate our model by showing qualitative and quantitative results on our collected database. We also compare to baseline detectors that use 2D constraints alone on the tasks of geometric primitive detection and part localization. We show improved performance on the part localization task.

## 2 Model for 3D cuboid localization

We represent the appearance of cuboids by a set of parts located at the corners of the cuboid and a set of internal edges. We enforce spatial consistency among the corners and edges by explicitly reasoning about its 3D shape. Let $I$ be the image and $p_i = (x_i, y_i)$ be the 2D image location of the $i$th corner on the cuboid. We define an undirected loopy graph $\mathcal{G} = (\mathcal{V}, \mathcal{E})$ over the corners of the cuboid, with vertices $\mathcal{V}$ and edges $\mathcal{E}$ connecting the corners of the cuboid. We illustrate our loopy graph layout in Figure 2(a). We define a scoring function associated with the corner locations in the image:

$$S(I, p) = \sum_{i \in \mathcal{V}} w_i^H \cdot \text{HOG}(I, p_i) + \sum_{ij \in \mathcal{E}} w_{ij}^D \cdot \text{Displacement}^{\text{2D}}(p_i, p_j)$$
$$+ \sum_{ij \in \mathcal{E}} w_{ij}^E \cdot \text{Edge}(I, p_i, p_j) + w^S \cdot \text{Shape}^{\text{3D}}(p) \quad (1)$$

where $\text{HOG}(I, p_i)$ is a HOG descriptor [4] computed at image location $p_i$ and $\text{Displacement}^{\text{2D}}(p_i, p_j) = -[(x_i - x_j)^2, x_i - x_j, (y_i - y_j)^2, y_i - y_j]$ is a 2D corner displacement term that is used in other pictorial parts-based models [7, 27]. By reasoning about the 3D shape, our model handles different 3D viewpoints and aspect ratios, as illustrated in Figure 2. Notice that our model is linear in the weights $w$. Moreover, the model is flexible as it adapts to the training data by automatically learning weights that measure the relative importance of the appearance and spatial terms. We define the Edge and Shape$^{\text{3D}}$ terms as follows.

Edge$(I, p_i, p_j)$: The internal edge information on cuboids is informative and provides a salient feature for the locations of the corners. For this, we include a term that models the appearance of the internal edges, which is illustrated in Figure 3. For adjacent corners on the cuboid, we identify the edge between the two corners and calculate the image evidence to support the existence of such an edge. Given the corner locations $p_i$ and $p_j$, we use Chamfer matching to align the straight line between the two corners to edges extracted from the image. We find image edges using Canny edge detection [3] and efficiently compute the distance of each pixel along the line segment to the nearest edge via the truncated distance transform. We use Bresenham's line algorithm [2] to efficiently find the 2D image locations on the line between the two points. The final edge term is the negative mean value of the Chamfer matching score for all pixels on the line. As there are usually 9 visible edges for a cuboid, we have 9 dimensions for the edge term.

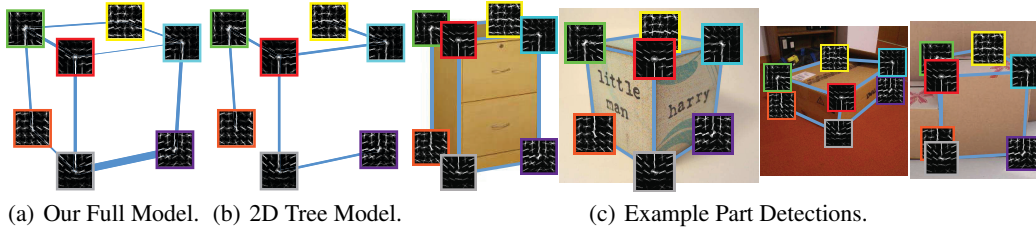

(a) Our Full Model. (b) 2D Tree Model.　　　　　(c) Example Part Detections.

Figure 2: Model visualization. Corresponding model parts are colored consistently in the figure. In (a) and (b) the displayed corner locations are the average 2D locations across all viewpoints and aspect ratios in our database. In (a) the edge thickness corresponds to the learned weight for the edge term. We can see that the bottom edge is significantly thicker, which indicates that it is informative for detection, possibly due to shadows and contact with a supporting plane.

Shape$^{3D}(p)$: The 3D shape of a cuboid constrains the layout of the parts and edges in the image. We propose to define a shape term that measures how well the configuration of corner locations respect the 3D shape. In other words, given the 2D locations $p$ of the corners, we define a term that tells us how likely this configuration of corner locations $p$ can be interpreted as the reprojection of a valid cuboid in 3D. When combined with the weights $w_S$, we get an overall evaluation of the goodness of the 2D locations with respect to the 3D shape. Let $\mathbf{X}$ (written in homogeneous coordinates) be the 3D points on the unit cube centered at the world origin. Then, $\mathbf{X}$ transforms as $\mathbf{x} = \mathbf{PLX}$, where $\mathbf{L}$ is a matrix that transforms the shape of the unit cube and $\mathbf{P}$ is a $3 \times 4$ camera matrix. For each corner, we use the other six 2D corner locations to estimate the product $\mathbf{PL}$ using camera resectioning [10]. The estimated matrix is used to predict the corner location. We use the negative L2 distance to the predicted corner location as a feature for the corner in our model. As we model 7 corners on the cuboid, there are 7 dimensions in the feature vector. When combined with the learned weights $w^S$ through dot-product, this represents a weighted reprojection error score.

## 2.1  Inference

Our goal is to find the 2D corner locations $p$ over the HOG grid of $I$ that maximizes the score given in Equation (1). Note that exact inference is hard due to the global shape term. Therefore, we propose a spanning tree approximation to the graph to obtain multiple initial solutions for possible corner locations. Then we adjust the corner locations using randomized simple hill climbing.

For the initialization, it is important for the computation to be efficient since we need to evaluate all possible detection windows in the image. Therefore, for simplicity and speed, we use a spanning tree $\mathcal{T}$ to approximate the full graph $\mathcal{G}$, as shown in Figure 2(b). In addition to the HOG feature as a unary term, we use a popular pairwise spring term along the edges of the tree to establish weak spatial constraints on the corners. We use the following scoring function for the initialization:

$$S_{\mathcal{T}}(I,p) = \sum_{i \in \mathcal{V}} w_i^H \cdot \text{HOG}(I, p_i) + \sum_{ij \in \mathcal{T}} w_{ij}^D \cdot \text{Displacement}^{2D}(p_i, p_j) \qquad (2)$$

Note that the model used for obtaining initial solutions is similar to [7, 27], which is only able to handle a fixed viewpoint and 2D aspect ratio. Nonetheless, we use it since it meets our speed requirement via dynamic programming and the distance transform [8].

With the tree approximation, we pick the top 1000 possible configurations of corner locations from each image and optimize our scoring function by adjusting the corner locations using randomized simple hill climbing. Given the initial corner locations for a single configuration, we iteratively choose a random corner $i$ with the goal of finding a new pixel location $\hat{p}_i$ that increases the scoring function given in Equation (1) while holding the other corner locations fixed. We compute the scores at neighboring pixel locations to the current setting $p_i$. We also consider the pixel location that the 3D rigid model predicts when estimated from the other corner locations. We randomly choose one of the locations and update $p_i$ if it yields a higher score. Otherwise, we choose another random corner and location. The algorithm terminates when no corner can reach a location that improves the score, which indicates that we have reached a local maxima.

During detection, since the edge and 3D shape terms are non-positive and the weights are constrained to be positive, this allows us to upper-bound the scoring function and quickly reject candidate loca-

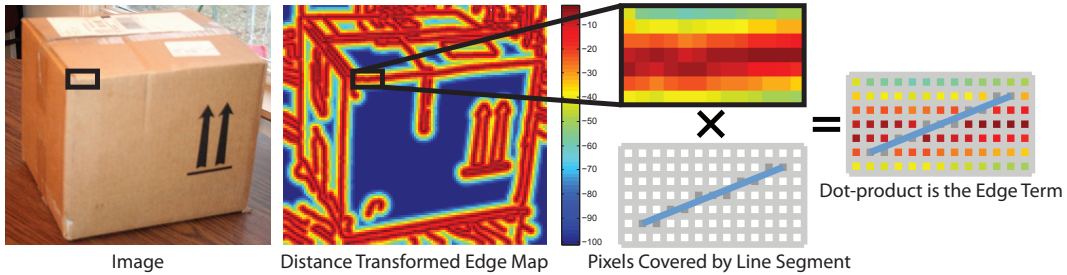

Image       Distance Transformed Edge Map       Pixels Covered by Line Segment

Dot-product is the Edge Term

Figure 3: Illustration of the edge term in our model. Given line endpoints, we compute a Chamfer matching score for pixels that lie on the line using the response from a Canny edge detector.

tions without evaluating the entire function. Also, since only one corner can change locations at each iteration, we can reuse the computed scoring function from previous iterations during hill climbing. Finally, we perform non-maximal suppression among the parts and then perform non-maximal suppression over the entire object to get the final detection result.

## 2.2 Learning

For learning, we first note that our scoring function in Equation (1) is linear in the weights $w$. This allows us to use existing structured prediction procedures for learning. To learn the weights, we adapt the structural SVM framework of [16]. Given positive training images with the 2D corner locations labeled $\{I_n, p_n\}$ and negative training images $\{I_n\}$, we wish to learn weights and bias term $\beta = (w^H, w^D, w^E, w^S, b)$ that minimizes the following structured prediction objective function:

$$\min_{\beta, \xi \geq 0} \quad \frac{1}{2}\beta \cdot \beta + C \sum_n \xi_n \tag{3}$$

$$\forall n \in \text{pos} \quad \beta \cdot \Phi\left(I_n, p_n\right) \geq 1 - \xi_n$$
$$\forall n \in \text{neg}, \forall p \in P \quad \beta \cdot \Phi\left(I_n, p\right) \leq -1 + \xi_n$$

where all appearance and spatial feature vectors are concatenated into the vector $\Phi(I_n, p)$ and $P$ is the set of all possible part locations. During training we constrain the weights $w^D, w^E, w^S \geq 0.0001$. We tried mining negatives from the wrong corner locations in the positive examples but found that it did not improve the performance. We also tried latent positive mining and empirically observed that it slightly helps. Since the latent positive mining helped, we also tried an offset compensation as post-processing to obtain the offset of corner locations introduced during latent positive mining. For this, we ran the trained detector on the training set to obtain the offsets and used the mean to compensate for the location changes. However, we observed empirically that it did not help performance.

## 2.3 Discussion

Sliding window object detectors typically use a root filter that covers the entire object [4] or a combination of root filter and part filters [7]. The use of a root filter is sufficient to capture the appearance for many object categories since they have canonical 3D viewpoints and aspect ratios. However, cuboids in general span a large number of object categories and do not have a consistent 3D viewpoint or aspect ratio. The diversity of 3D viewpoints and aspect ratios causes dramatic changes in the root filter response. However, we have observed that the responses for the part filters are less affected.

Moreover, we argue that a purely view-based approach that trains separate models for the different viewpoints and aspect ratios may not capture well this diversity. For example, such a strategy would require dividing the training data to train each model. In contrast, we train our model for all 3D viewpoints and aspect ratios. We illustrate this in Figure 2, where detected parts are colored consistently in the figure. As our model handles different viewpoints and aspect ratios, we are able to make use of the entire database during training.

Due to the diversity of cuboid appearance, our model is designed to capture the most salient features, namely the corners and edges. While the corners and edges may be occluded (e.g. by self-occlusion,

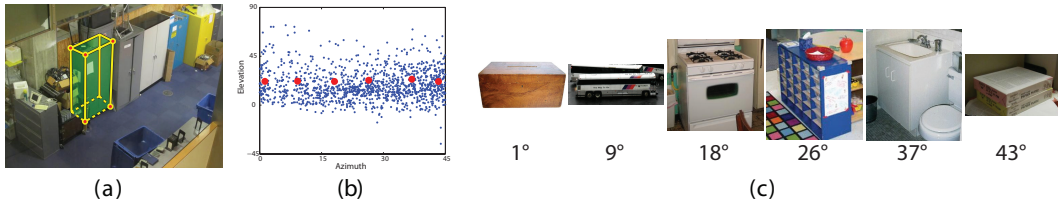

|  (a)  |  (b)  |  (c)  |

Figure 4: Illustration of the labeling tool and 3D viewpoint statistics. (a) A cuboid being labeled through the tool. A projection of the cuboid model is overlaid on the image and the user must select and drag anchor points to their corresponding location in the image. (b) Scatter plot of 3D azimuth and elevation angles for annotated cuboids with zenith angle close zero. We perform an image left/right swap to limit the rotation range. (c) Crops of cuboids at different azimuth angles for a fixed elevation, with the shown examples marked as red points in the scatter plot of (b).

other objects in front, or cropping), for now we do not handle these cases explicitly in our model. Furthermore, we do not make use of other appearance cues, such as the appearance within the cuboid faces, since they have a larger variation across the object categories (e.g. dice and fire alarm trigger) and may not generalize as well. We also take into account the tractability of our model as adding additional appearance cues will increase the complexity of our model and the detector needs to be evaluated over a large number of possible sliding windows in an image.

Compared with recent approaches that detect cuboids by reasoning about the shape of the entire scene [9, 11, 12, 17, 19, 29], one of the key differences is that we detect cuboids directly without consideration of the global scene geometry. These prior approaches rely heavily on the assumption that the camera is located *inside* a cuboid-like room and held at human height, with the parameters of the room cuboid inferred through vanishing points based on a Manhattan world assumption. Therefore, they cannot handle outdoor scenes or close-up snapshots of an object (e.g. the boxes on a shelf in row 1, column 3 of Figure 6). As our detector is agnostic to the scene geometry, we are able to detect cuboids even when these assumptions are violated.

While previous approaches reason over rigid cuboids, our model is flexible in that it can adapt to deformations of the 3D shape. We observe that not all cuboid-like objects are perfect cuboids in practice. Deformations of the shape may arise due to the design of the object (e.g. the printer in Figure 1), natural deformation or degradation of the object (e.g. a cardboard box), or a global transformation of the image (e.g. camera radial distortion). We argue that modeling the deformations is important in practice since a violation of the rigid constraints may make a 3D reconstruction-based approach numerically unstable. In our approach, we model the 3D deformation and allow the structural SVM to learn based on the training data how to weight the importance of the 3D shape term. Moreover, a rigid shape requires a perfect 3D reconstruction and it is usually done with non-linear optimization [17], which is expensive to compute and becomes impractical in an exhaustive sliding-window search in order to maintain a high recall rate. With our approach, if a rigid cuboid is needed, we can recover the 3D shape parameters via camera resectioning, as shown in Figure 9.

## 3   Database of 3D cuboids

To develop and evaluate any models for 3D cuboid detection in real-world environments, it is necessary to have a large database of images depicting everyday scenes with 3D cuboids labeled. In this work, we seek to build a database by manually labeling point correspondences between images and 3D cuboids. We have built a labeling tool that allows a user to select and drag key points on a projected 3D cuboid model to its corresponding location in the image. This is similar to existing tools, such as Google building maker [14], which has been used to build 3D models of buildings for maps. Figure 4(a) shows a screenshot of our tool. For the database, we have harvested images from four sources: (i) a subset of the SUN database [25], which contains images depicting a large variety of different scene categories, (ii) ImageNet synsets [5] with objects having one or more 3D cuboids depicted, (iii) images returned from an Internet search using keywords for objects that are wholly or partially described by 3D cuboids, and (iv) a set of images that we manually collected from our personal photographs. Given the corner correspondences, the parameters for the 3D cuboids and camera are estimated. The cuboid and camera parameters are estimated up to a similarity transformation via camera resectioning using Levenberg-Marquardt optimization [10].

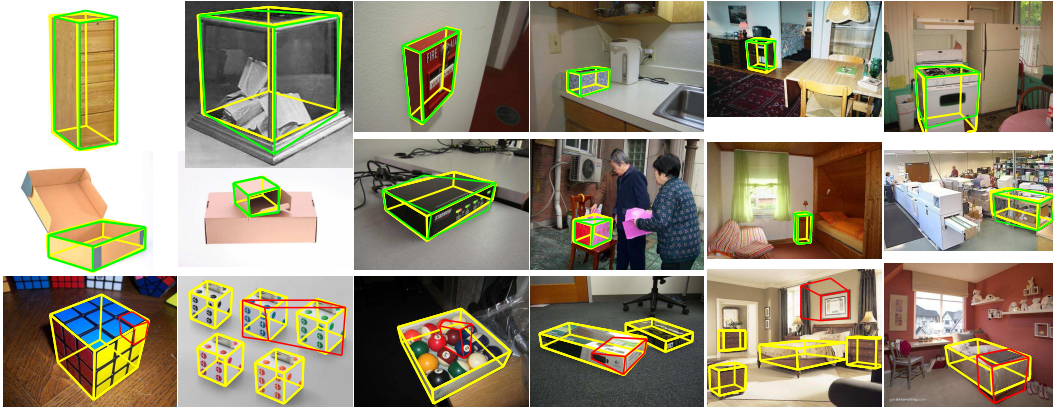

Figure 5: Single top 3D cuboid detection in each image. Yellow: ground truth, green: correct detection, red: false alarm. Bottom row - false positives. The false positives tend to occur when a part fires on a "cuboid-like" corner region (e.g. row 3, column 5) or finds a smaller cuboid (e.g. the Rubik's cube depicted in row 3, column 1).

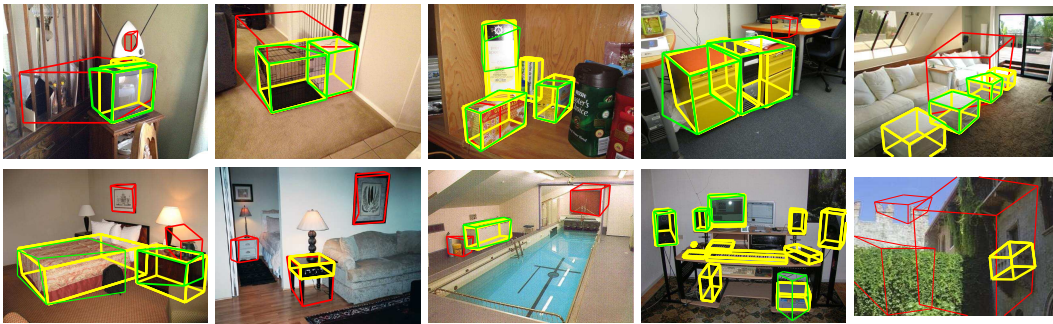

Figure 6: All 3D cuboid detections above a fixed threshold in each image. Notice that our model is able to detect the presence of multiple cuboids in an image (e.g. row 1, columns 2-5) and handles partial occlusions (e.g. row 1, column 4), small objects, and a range of 3D viewpoints, aspect ratios, and object classes. Moreover, the depicted scenes have varying amount of clutter. Yellow - ground truth. Green - correct prediction. Red - false positive. Line thickness corresponds to detector confidence.

For our database, we have 785 images with 1269 cuboids annotated. We have also collected a negative set containing 2746 images that do contain any cuboid like objects. We perform an image left/right swap to limit the rotation range. As a result, the min/max azimuth, elevation, and zenith angles are 0/45, -90/90, -180/180 degrees respectively. In Figure 4(b) we show a scatter plot of the azimuth and elevation angles for all of the labeled cuboids with zenith angle close to zero. Notice that the cuboids cover a large range of azimuth angles for elevation angles between 0 (frontal view) and 45 degrees. We also show a number of cropped examples for a fixed elevation angle in Figure 4(c), with their corresponding azimuth angles indicated by the red points in the scatter plot. Figure 8(c) shows the distribution of objects from the SUN database [25] that overlap with our cuboids (there are 326 objects total from 114 unique classes). Compared with [12], our database covers a larger set of object and scene categories, with images focusing on both objects and scenes (all images in [12] are indoor scene images). Moreover, we annotate objects closely resembling a 3D cuboid (in [12] there are many non-cuboids that are annotated with a bounding cuboid) and overall our cuboids are more accurately labeled.

## 4 Evaluation

In this section we show qualitative results of our model on the 3D cuboids database and report quantitative results on two tasks: (i) 3D cuboid detection and (ii) corner localization accuracy. For training and testing, we randomly split equally the positive and negative images. As discussed in Section 3, there is rotational symmetry in the 3D cuboids. During training, we allow the image

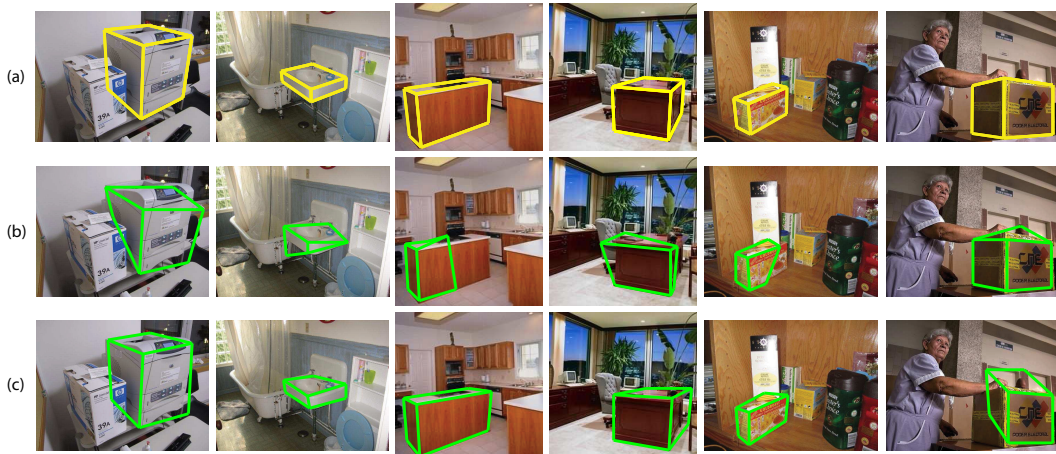

Figure 7: Corner localization comparison for detected geometric primitives. (a) Input image and ground truth annotation. (b) 2D tree-based initialization. (c) Our full model. Notice that our model is able to better localize cuboid corners over the baseline 2D tree-based model, which corresponds to 2D parts-based models used in object detection and articulated pose estimation [7, 27]. The last column shows a failure case where a part fires on a "cuboid-like" corner region in the image.

to mirror left-right and orient the 3D cuboid to minimize the variation in rotational angle. During testing, we run the detector on left-right mirrors of the image and select the output at each location with the highest detector response. For the parts we extract HOG features [4] in a window centered at each corner with scale of 10% of the object bounding box size. Figure 5 shows the single top cuboid detection in each image and Figure 6 shows all of the most confident detections in the image. Notice that our model is able to handle partial occlusions (e.g. row 1, column 4 of Figure 6), small objects, and a range of 3D viewpoints, aspect ratios, and object classes. Moreover, the depicted scenes have varying amount of clutter. We note that our model fails when a corner fires on a "cuboid-like" corner region (e.g. row 3, column 5 of Figure 5).

We compare the various components of our model against two baseline approaches. The first baseline is a root HOG template [4] trained over the appearance within a bounding box covering the entire object. A single model using the root HOG template is trained for all viewpoints and aspect ratios. During detection, output corner locations corresponding to the average training corner locations relative to the bounding boxes are returned. The second baseline is the 2D tree-based approximation of Equation (2), which corresponds to existing 2D parts models used in object detection and articulated pose estimation [7, 27]. Figure 7 shows a qualitative comparison of our model against the 2D tree-based model. Notice that our model localizes well and often provides a tighter fit to the image data than the baseline model.

We evaluate geometric primitive detection accuracy using the bounding box overlap criteria in the Pascal VOC [6]. We report precision recall in Figure 8(a). We have observed that all of the corner-based models achieve almost identical detection accuracy across all recall levels, and out-perform the root HOG template detector [4]. This is expected as we initialize our full model with the output of the 2D tree-based model and it generally does not drift too far from this initialization. This in effect does not allow us to detect additional cuboids but allows for better part localization.

In addition to detection accuracy, we also measure corner localization accuracy for correctly detected examples for a given model. A corner is deemed correct if its predicted image location is within $t$ pixels of the ground truth corner location. We set $t$ to be 15% of the square root of the area of the ground truth bounding box for the object. The reported trends in the corner localization performance hold for nearby values of $t$. In Figure 8 we plot corner localization accuracy as a function of recall and compare our model against the two baselines. Moreover, we report performance when either the edge term or the 3D shape term is omitted from our model. Notice that our full model out-performs the other baselines. Also, the additional edge and 3D shape terms provide a gain in performance over using the appearance and 2D spatial terms alone. The edge term provides a slightly larger gain in performance over the 3D shape term, but when integrated together consistently provides the best performance on our database.

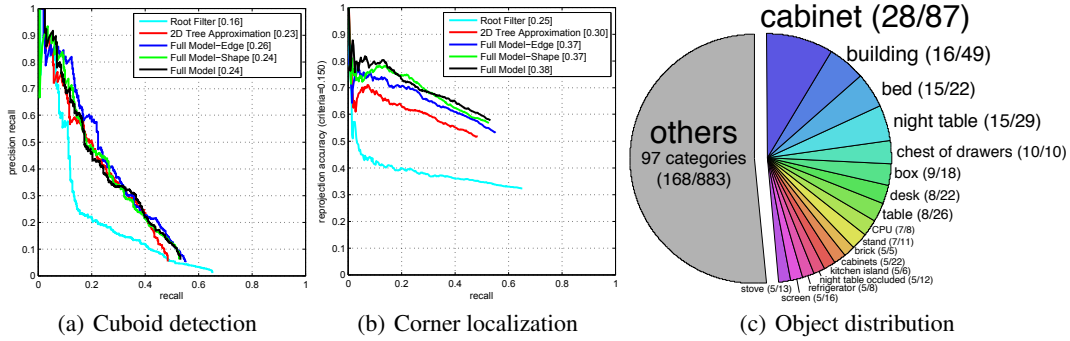

(a) Cuboid detection   (b) Corner localization   (c) Object distribution

Figure 8: Cuboid detection (precision vs. recall) and corner localization accuracy (accuracy vs. recall). The area under the curve is reported in the plot legends. Notice that all of the corner-based models achieve almost identical detection accuracy across all recall levels and out-perform the root HOG template detector [4]. For the task of corner localization, our full model out-performs the two baseline detectors or when either the Edge or Shape$^{3D}$ terms are omitted from our model. (c) Distribution of objects from the SUN database [25] that overlap with our cuboids. There are 326 objects total from 114 unique classes. The first number within the parentheses indicates the number of instances in each object category that overlaps with a labeled cuboid, while the second number is the total number of labeled instances for the object category within our dataset.

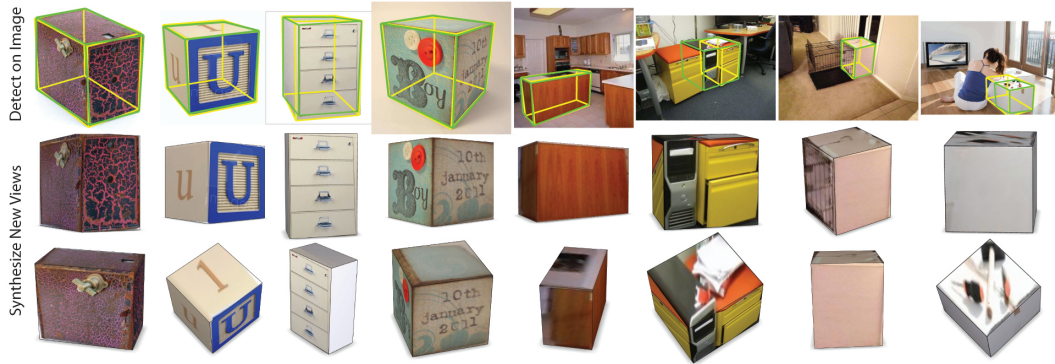

Figure 9: Detected cuboids and subsequent synthesized new views via camera resectioning.

## 5   Conclusion

We have introduced a novel model that detects 3D cuboids and localizes their corners in single-view images. Our 3D cuboid detector makes use of both corner and edge information. Moreover, we have constructed a dataset with ground truth cuboid annotations. Our detector handles different 3D viewpoints and aspect ratios and, in contrast to recent approaches for 3D cuboid detection, does not make any assumptions about the scene geometry and allows for deformation of the 3D cuboid shape. As HOG is not invariant to viewpoint, we believe that part mixtures would allow the model to be invariant to viewpoint. We believe our approach extends to other shapes, such as cylinders and pyramids. Our work raises a number of (long-standing) issues that would be interesting to address. For instance, which objects can be described by one or more geometric primitives and how to best represent the compositionality of objects in general? By detecting geometric primitives, what applications and systems can be developed to exploit this? Our dataset and source code is publicly available at the project webpage: `http://SUNprimitive.csail.mit.edu`.

**Acknowledgments:** Jianxiong Xiao is supported by Google U.S./Canada Ph.D. Fellowship in Computer Vision. Bryan Russell was funded by the Intel Science and Technology Center for Pervasive Computing (ISTC-PC). This work is funded by ONR MURI N000141010933 and NSF Career Award No. 0747120 to Antonio Torralba.

# References

[1] I. Biederman. Recognition by components: a theory of human image interpretation. *Pyschological review*, 94:115–147, 1987.

[2] J. E. Bresenham. Algorithm for computer control of a digital plotter. *IBM Systems Journal*, 4(1):25–30, 1965.

[3] J. F. Canny. A computational approach to edge detection. *IEEE PAMI*, 8(6):679–698, 1986.

[4] N. Dalal and B. Triggs. Histograms of Oriented Gradients for Human Detection. In *CVPR*, 2005.

[5] J. Deng, W. Dong, R. Socher, L.-J. Li, K. Li, and L. Fei-Fei. ImageNet: A large-scale hierarchical image database. In *CVPR*, 2009.

[6] M. Everingham, L. Van Gool, C. K. I. Williams, J. Winn, and A. Zisserman. The Pascal visual object classes (VOC) challenge. *IJCV*, 88(2):303–338, 2010.

[7] P. Felzenszwalb, R. Girshick, D. McAllester, and D. Ramanan. Object detection with discriminatively trained part based models. *IEEE PAMI*, 32(9), 2010.

[8] P. Felzenszwalb and D. Huttenlocher. Pictorial structures for object recognition. *IJCV*, 61(1), 2005.

[9] A. Gupta, S. Satkin, A. A. Efros, and M. Hebert. From 3d scene geometry to human workspace. In *CVPR*, 2011.

[10] R. I. Hartley and A. Zisserman. *Multiple View Geometry in Computer Vision*. Cambridge University Press, ISBN: 0521540518, second edition, 2004.

[11] V. Hedau, D. Hoiem, and D. Forsyth. Thinking inside the box: Using appearance models and context based on room geometry. In *ECCV*, 2010.

[12] V. Hedau, D. Hoiem, and D. Forsyth. Recovering free space of indoor scenes from a single image. In *CVPR*, 2012.

[13] D. Hoiem, A. Efros, and M. Hebert. Geometric context from a single image. In *ICCV*, 2005.

[14] http://sketchup.google.com, 2012.

[15] K. Ikeuchi and T. Suehiro. Toward an assembly plan from observation: Task recognition with polyhedral objects. In *Robotics and Automation*, 1994.

[16] T. Joachims, T. Finley, and C.-N. J. Yu. Cutting-plane training of structural svms. *Machine Learning*, 77(1), 2009.

[17] D. C. Lee, A. Gupta, M. Hebert, and T. Kanade. Estimating spatial layout of rooms using volumetric reasoning about objects and surfaces. In *NIPS*, 2010.

[18] J. L. Mundy. Object recognition in the geometric era: A retrospective. In *Toward Category-Level Object Recognition, volume 4170 of Lecture Notes in Computer Science*, pages 3–29. Springer, 2006.

[19] L. D. Pero, J. C. Bowdish, D. Fried, B. D. Kermgard, E. L. Hartley, and K. Barnard. Bayesian geometric modelling of indoor scenes. In *CVPR*, 2012.

[20] L. Roberts. Machine perception of 3-d solids. In *PhD. Thesis*, 1965.

[21] H. Wang, S. Gould, and D. Koller. Discriminative learning with latent variables for cluttered indoor scene understanding. In *ECCV*, 2010.

[22] J. Xiao, T. Fang, P. Tan, P. Zhao, E. Ofek, and L. Quan. Image-based façade modeling. In *SIGGRAPH Asia*, 2008.

[23] J. Xiao, T. Fang, P. Zhao, M. Lhuillier, and L. Quan. Image-based street-side city modeling. In *SIGGRAPH Asia*, 2009.

[24] J. Xiao and Y. Furukawa. Reconstructing the world's museums. In *ECCV*, 2012.

[25] J. Xiao, J. Hays, K. Ehinger, A. Oliva, and A. Torralba. SUN database: Large-scale scene recognition from abbey to zoo. In *CVPR*, 2010.

[26] J. Xiao, B. C. Russell, J. Hays, K. A. Ehinger, A. Oliva, and A. Torralba. Basic level scene understanding: From labels to structure and beyond. In *SIGGRAPH Asia*, 2012.

[27] Y. Yang and D. Ramanan. Articulated pose estimation using flexible mixtures of parts. In *CVPR*, 2011.

[28] S. Yu, H. Zhang, and J. Malik. Inferring spatial layout from a single image via depth-ordered grouping. In *IEEE Workshop on Perceptual Organization in Computer Vision*, 2008.

[29] Y. Zhao and S.-C. Zhu. Image parsing with stochastic scene grammar. In *NIPS*. 2011.

